# The Gamma MLP for Speech Phoneme Recognition

**Steve Lawrence**,* **Ah Chung Tsoi, Andrew D. Back**
{lawrence,act,back}@elec.uq.edu.au

Department of Electrical and Computer Engineering
University of Queensland
St. Lucia Qld 4072 Australia

## Abstract

We define a Gamma multi-layer perceptron (MLP) as an MLP with the usual synaptic weights replaced by gamma filters (as proposed by de Vries and Principe (de Vries and Principe, 1992)) and associated gain terms throughout all layers. We derive gradient descent update equations and apply the model to the recognition of speech phonemes. We find that both the inclusion of gamma filters in all layers, and the inclusion of synaptic gains, improves the performance of the Gamma MLP. We compare the Gamma MLP with TDNN, Back-Tsoi FIR MLP, and Back-Tsoi IIR MLP architectures, and a local approximation scheme. We find that the Gamma MLP results in an substantial reduction in error rates.

## 1 INTRODUCTION

### 1.1 THE GAMMA FILTER

Infinite Impulse Response (IIR) filters have a significant advantage over Finite Impulse Response (FIR) filters in signal processing: the length of the impulse response is uncoupled from the number of filter parameters. The length of the impulse response is related to the memory depth of a system, and hence IIR filters allow a greater memory depth than FIR filters of the same order. However, IIR filters are

not widely used in practical adaptive signal processing. This may be attributed to the fact that a) there could be instability during training and b) the gradient descent training procedures are not guaranteed to locate the global optimum in the possibly non-convex error surface (Shynk, 1989).

De Vries and Principe proposed using gamma filters (de Vries and Principe, 1992), a special case of IIR filters, at the input to an otherwise standard MLP. The gamma filter is designed to retain the uncoupling of memory depth to the number of parameters provided by IIR filters, but to have simple stability conditions.

The output of a neuron in a multi-layer perceptron is computed using[1] $y_k^l = f\left(\sum_{i=0}^{N_l-1} w_{ki}^l y_i^{l-1}\right)$. De Vries and Principe consider adding short term memory with delays: $y_k^l = f\left(\sum_{i=0}^{N_l-1} \sum_{j=0}^{K} g_{kij}^l(t-j) y_i^{l-1}(t-j)\right)$ where $g_{kij}^l = \frac{\mu_{ki}^l}{(j-1)!} t^{j-1} e^{-\mu_{ki}^l t}$   $j = 1, ..., K$ . The depth of the memory is controlled by $\mu$, and $K$ is the order of the filter. For the discrete time case, we obtain the recurrence relation: $z_0(t) = x(t)$ and $z_j(t) = (1-\mu)z_j(t-1) + \mu z_{j-1}(t-1)$ for $j = 1, ..., K$. In this form, the gamma filter can be interpreted as a cascaded series of filter modules, where each module is a first order IIR filter with the transfer function $\frac{\mu}{q-(1-\mu)}$, where $qz_j(t) \stackrel{\triangle}{=} z_j(t+1)$. We have a filter with $K$ poles, all located at $1 - \mu$. Thus, the gamma filter may be considered as a low pass filter for $\mu < 1$. The value of $\mu$ can be fixed, or it can be adapted during training.

## 2   NETWORK MODELS

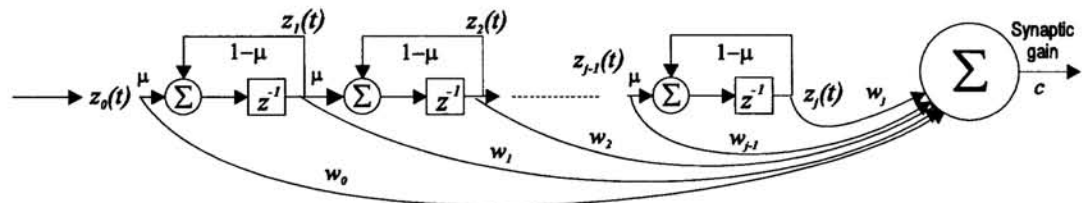

Figure 1: A gamma filter synapse with an associated gain term 'c'.

We have defined a gamma MLP as a multi-layer perceptron where every synapse contains a gamma filter and a gain term, as shown in figure 1. The motivation behind the inclusion of the gain term is discussed later. A separate $\mu$ parameter is used for each filter. Update equations are derived in a manner analogous to the standard MLP and can be found in Appendix A. The model is defined as follows.

**Definition 1** A Gamma MLP with L layers excluding the input layer $(0, 1, ..., L)$, gamma filters of order $K$, and $N_0, N_1, ..., N_L$ neurons per layer, is defined as

$$
\begin{aligned}
y_k^l(t) &= f\left(x_k^l(t)\right) \\
x_k^l(t) &= \sum_{i=0}^{N_{l-1}} c_{ki}^l(t) \sum_{j=0}^{K} w_{kij}^l(t) z_{kij}^l(t) \\
z_{kij}^l(t) &= (1 - \mu_{ki}^l(t)) z_{kij}^l(t-1) + \mu_{ki}^l(t) z_{ki(j-1)}^l(t-1) \quad 1 \le j \le K \\
z_{kij}^l(t) &= y_i^{l-1}(t) \quad\quad\quad\quad\quad\quad\quad\quad\quad\quad\quad\quad\quad\quad j = 0
\end{aligned}
\tag{1}
$$

where $y(t)$ = neuron output, $c_{ki}^l$ = synaptic gain, $f(\alpha) = \frac{e^{\alpha/2} - e^{-\alpha/2}}{e^{\alpha/2} + e^{-\alpha/2}}$, $k = 1, 2, ..., N_l$(neuron index), $l = 0, 1, ..., L$(layer), and $z_{kij}^l|_{i=0} = 1, w_{kij}^l|_{i=0, j \ne 0} = 0, c_{kij}^l|_{i=0} = 1$(bias).

$\square$

For comparison purposes, we have used the TDNN (Time Delay Neural Network) architecture[2], the Back-Tsoi FIR[3] and IIR MLP architectures (Back and Tsoi, 1991a) where every synapse contains an FIR or IIR filter and a gain term, and the local approximation algorithm used by Casdagli (k-NN LA) (Casdagli, 1991)[4]. The Gamma MLP is a special case of the IIR MLP.

## 3 TASK

### 3.1 MOTIVATION

Accurate speech recognition requires models which can account for a high degree of variability in the data. Large amounts of data may be available but it may be impractical to use all of the information in standard neural network models.

*Hypothesis*: As the complexity of a problem increases (higher dimensionality, greater variety of training data), the error surface of a neural network becomes more complex. It may contain a number of local minima[5] many of which may be much worse than the global minimum. The training (parameter estimation) algorithms become "stuck" in local minima which may be increasingly poor compared to the global optimum. The problem suffers from the so called "curse of dimensionality" and the

difficulty in optimizing a function with limited control over the nature of the error surface.

We can identify two main reasons why the application of the Gamma MLP may be superior to the standard TDNN for speech recognition: a) the gamma filtering operation allows consideration of the input data using different time resolutions and can account for more past history of the signal which can only be accounted for in an FIR or TDNN system by increasing the dimensionality of the model, and b) the low pass filtering nature of the gamma filter may create a smoother function approximation task, and therefore a smoother error surface for gradient descent[6].

## 3.2  TASK DETAILS

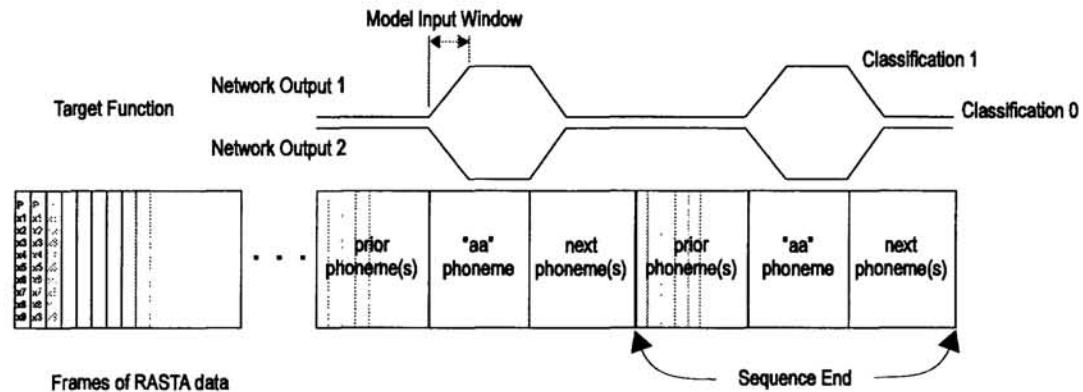

Figure 2: PLP input data format and the corresponding network target functions for the phoneme "aa".

Our data consists of phonemes extracted from the TIMIT database and organized as a number of sequences as shown in figure 2 (example for the phoneme "aa"). One model is trained for each phoneme. Note that the phonemes are classified in context, with a number of different contexts, and that the surrounding phonemes are labelled only as not belonging to the target phoneme class. Raw speech data was pre-processed into a sequence of frames using the RASTA-PLP v2.0 software[7]. We used the default options for PLP analysis. The analysis window (frame) was 20 ms. Each succeeding frame overlaps with the preceding frame by 10 ms. 9 PLP coefficients together with the signal power are extracted and used as features describing each frame of data. Phonemes used in the current tests were the vowel "aa" and the fricative "s". The phonemes were extracted from speakers coming from the same demographic region in the TIMIT database. Multiple speakers were used and the speakers used in the test set were not contained in the training set. The training set contained 4000 frames, where each phoneme is roughly 10 frames. The test set contained 2000 frames, and an additional validation set containing 2000 frames was used to control generalization.

## 4 RESULTS

Two outputs were used in the neural networks as shown by the target functions in figure 2, corresponding to the phoneme being present or not. A confidence criterion was used: $y_{max} \times (y_{max} - y_{min})$ (for softmax outputs). The initial learning rate was 0.1, 10 hidden nodes were used, FIR and Gamma orders were 5 (6 taps), the TDNN and k-NN models had an input window of 6 steps in time, the *tanh* activation function was used, target outputs were scaled between -0.8 and 0.8, stochastic update was used, and initial weights were chosen from a set of candidates based on training set performance. The learning rate was varied over time according to the schedule:

$$\eta = \eta_0 / \left( \frac{n}{N/2} + \frac{c_1}{max\left(1,(c_1 - \frac{max(0,c_1(n-c_2N))}{(1-c_2)N})\right)} \right)$$ where $\eta$ = learning rate, $\eta_0$ = initial learning rate, $N$ = total epochs, $n$ = current epoch, $c_1 = 50$, $c_2 = 0.65$. This is similar to the schedule proposed in (Darken and Moody, 1991) with an additional term to decrease the learning rate towards zero over the final epochs[8].

| Train Error % | 2-NN | 5-NN | 1st layer | | All layers | | Gains, 1st layer | | Gains, all layers | |
|---|---|---|---|---|---|---|---|---|---|---|
| FIR MLP | | | 17.6 | 0.43 | 14.5 | 1.5 | 27.2 | 0.59 | 40.9 | 19.8 |
| Gamma MLP | | | 7.78 | 0.39 | 5.73 | 0.88 | 6.07 | 0.12 | 5.63 | 1.68 |
| TDNN | | | | | | | | | 14.4 | 0.86 |
| k-NN LA | 0 | 0 | | | | | | | | |

| Test Error % | 2-NN | 5-NN | 1st layer | | All layers | | Gains, 1st layer | | Gains, all layers | |
|---|---|---|---|---|---|---|---|---|---|---|
| FIR MLP | | | 22.2 | 0.97 | 20.4 | 0.61 | 29 | 0.14 | 41 | 21 |
| Gamma MLP | | | 14.7 | 0.16 | 13.5 | 0.33 | 12.8 | 1.0 | 12.7 | 0.50 |
| TDNN | | | | | | | | | 24.5 | 0.68 |
| k-NN LA | 31 | 28.4 | | | | | | | | |

| Test False +ve | 2-NN | 5-NN | 1st layer | | All layers | | Gains, 1st layer | | Gains, all layers | |
|---|---|---|---|---|---|---|---|---|---|---|
| FIR MLP | | | 13.5 | 0.67 | 11.4 | 2.0 | 4.5 | 0.77 | 31.3 | 49.0 |
| Gamma MLP | | | 7.94 | 0.45 | 7.01 | 0.47 | 6.83 | 0.34 | 8.05 | 1.8 |
| TDNN | | | | | | | | | 13 | 0.27 |
| k-NN LA | 22.6 | 17.4 | | | | | | | | |

| Test False -ve | 2-NN | 5-NN | 1st layer | | All layers | | Gains, 1st layer | | Gains, all layers | |
|---|---|---|---|---|---|---|---|---|---|---|
| FIR MLP | | | 44.9 | 2.6 | 44.1 | 5.6 | 92.9 | 2.4 | 66.4 | 53 |
| Gamma MLP | | | 32.2 | 1.2 | 30.4 | 2.2 | 28.4 | 2.8 | 24.7 | 4.4 |
| TDNN | | | | | | | | | 54.6 | 1.8 |
| k-NN LA | 53 | 56.8 | | | | | | | | |

Table 1: Results comparing the architectures and the use of filters in all layers and synaptic gains for the FIR and Gamma MLP models. The NMSE is followed by the standard deviation. The TDNN results are listed under an arbitrary column heading (gains and 1st layer/all layers does not apply).

The results of the simulations are shown in table 1[9]. Each result represents an average over four simulations with different random seeds - the standard deviation of the four individual results is also shown. The FIR and Gamma MLP networks have been tested both with and without synaptic gains, and with and without filters in the output layer synapses. These results are for the models trained on the "s" phoneme, results for the "aa" phoneme exhibit the same trend. "Test false negative" is probably the most important result here, and is shown graphically in figure 3. This is the percentage of times a true classification (ie. the current

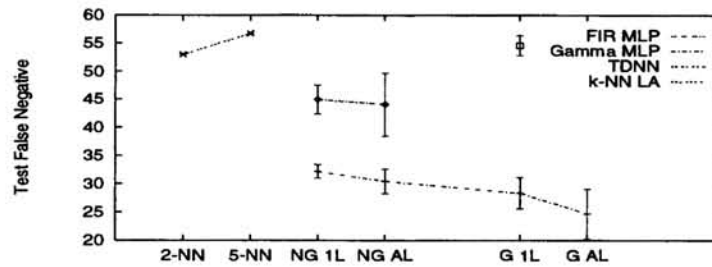

Figure 3: Percentage of false negative classifications on the test set. NG=No gains, G=Gains, 1L=filters in the first layer only, AL=filters in all layers. The error bars show plus and minus one standard deviation. The synaptic gains case for the FIR MLP is not shown as the poor performance compresses the remainder of the graph. Top to bottom, the lines correspond to: k-NN LA (left), TDNN, FIR MLP, and Gamma MLP.

phoneme is present) is incorrectly reported as false. From the table we can see that the Gamma MLP performs significantly better than the FIR MLP or standard TDNN models for this problem. Synaptic gains and gamma filters in all layers improve the performance of the Gamma MLP, while the inclusion of synaptic gains presented difficulty for the FIR MLP. Results for the IIR MLP are not shown - we have been unable to obtain significant convergence[10]. We investigated values of k not listed in the table for the k-NN LA model, but it performed poorly in all cases.

## 5   CONCLUSIONS

We have defined a Gamma MLP as an MLP with gamma filters and gain terms in every synapse. We have shown that the model performs significantly better on our speech phoneme recognition problem when compared to TDNN, Back-Tsoi FIR and IIR MLP architectures, and Casdagli's local approximation model. The percentage of times a phoneme is present but not recognized for the Gamma MLP was 44% lower than the closest competitor, the Back-Tsoi FIR MLP model.

The inclusion of gamma filters in all layers and the inclusion of synaptic gains improved the performance of the Gamma MLP. The improvement due to the inclusion of synaptic gains may be considered non-intuitive to many - we are adding degrees of freedom, but no additional representational power. The error surface will be different in each case, and the results indicate that the surface for the synaptic gains case is more amenable to gradient descent. One view of the situation is seen by Back & Tsoi with their FIR and IIR MLP networks (Back and Tsoi, 1991b): From a signal processing perspective the response of each synapse is determined by pole-zero positions. With no synaptic gains, the weights determine both the static gain and the pole-zero positions of the synapses. In an experimental analysis performed by Back & Tsoi it was observed that some synapses devoted themselves to model-

ing the dynamics of the system in question, while others "sacrificed" themselves to provide the necessary static gains[11] to construct the required nonlinearity.

## APPENDIX A: GAMMA MLP UPDATE EQUATIONS

$$\Delta w^l_{kij}(t) = -\eta \frac{\partial J(t)}{\partial w^l_{kij}(t)} = \eta \delta^l_k(t) c^l_{ki}(t) z^l_{kij}(t) \tag{2}$$

$$\Delta c^l_{ki}(t) = -\eta \frac{\partial J(t)}{\partial c^l_{ki}(t)} = \eta \delta^l_k(t) \sum_{j=0}^{K} w^l_{kij}(t) z^l_{kij}(t) \tag{3}$$

$$\Delta \mu^l_{ki}(t) = -\eta \frac{\partial J(t)}{\partial \mu^l_{ki}(t)} = \eta \delta^l_k(t) c^l_{ki}(t) \sum_{j=0}^{K} w^l_{kij}(t) \alpha^l_{kij}(t) \tag{4}$$

$$
\begin{aligned}
\alpha^l_{kij}(t) &= 0 & j=0 \\
&= (1-\mu^l_{ki}(t))\alpha^l_{kij}(t-1) + \mu^l_{ki}(t)\alpha^l_{ki(j-1)}(t-1) \\
&\quad + z^l_{ki(j-1)}(t-1) - z^l_{kij}(t-1) & 1 \le j \le K
\end{aligned} \tag{5}
$$

$$
\begin{aligned}
\delta^l_k(t) &= -\frac{\partial J(t)}{\partial x^l_k(t)} \\
&= e_k(t) f'(x^l_k(t)) & l=L \\
&= f'\left(x^l_k(t)\right) \sum_{p=1}^{N_{l+1}} \delta^{l+1}_p(t) c^{l+1}_{pk}(t) \sum_{j=0}^{K} w^{l+1}_{pkj}(t) \beta^{l+1}_{pkj}(t) & 1 \le j \le K
\end{aligned} \tag{6}
$$

$$
\begin{aligned}
\beta^l_{pkj}(t) &= 1 & j=0 \\
&= (1-\mu^l_{pk}(t))\beta^l_{pkj}(t-1) + \mu^l_{pk}(t)\beta^l_{pk(j-1)}(t-1) & 1 \le j \le K
\end{aligned} \tag{7}
$$

## Acknowledgments

This work has been partially supported by the Australian Research Council (ACT and ADB) and the Australian Telecommunications and Electronics Research Board (SL).

## Footnotes

*http://www.neci.nj.nec.com/homepages/lawrence

[1]where $y_k^l$ is the output of neuron $k$ in layer $l$, $N_l$ is the number of neurons in layer $l$, $w_{ki}^l$ is the weight connecting neuron $k$ in layer $l$ to neuron $i$ in layer $l-1$, $y_0^l = 1$ (bias), and $f$ is commonly a sigmoid function.

[2]We use TDNN to refer to an MLP with a time window of inputs, not the replicated architecture introduced by Lang (Lang et al., 1990).

[3]We distinguish the Back-Tsoi FIR network from the Wan FIR network in that the Wan architecture has no synaptic gains, and the update algorithms are different. The Back-Tsoi update algorithm has provided better convergence in previous experiments.

[4]Casdagli created an affine model of the following form for each test pattern: $y^j = \alpha_0 + \sum_{i=1}^{n} \alpha_i x_i^j$, where $k$ is the number of neighbors, $j = 1, ..., k$, and $n$ is the input dimension. The resulting model is used to find $y$ for the test pattern.

[5]We note that it can be difficult to distinguish a true local minimum from a long plateau in the standard backpropagation algorithm.

[6]If we consider a very simple network and derive the relationship of the smoothness of the required function approximation to the smoothness of the error surface this statement appears to be valid. However, it is difficult to show a direct relationship for general networks.

[7]Obtained from ftp://ftp.icsi.berkeley.edu/pub/speech/rasta2.0.tar.Z.

[8]Without this term we have encountered considerable parameter fluctuation over the last epoch.

[9]NMSE $= \sum_{k=1}^{N} (d(k) - y(k))^2 / \left( \sum_{k=1}^{N} \left( d(k) - \left( \sum_{k=1}^{N} d(k) \right) /N \right)^2 \right) /N.$

[10]Theoretically, the IIR MLP model is the most powerful model used here. Though it is prone to stability problems, the stability of the model can and was controlled in the simulations performed here (basically, by reflecting poles that move outside the unit circle back inside). The most obvious hypothesis for the difficulty in training the model is related to the error surface and the nature of gradient descent. We expect the error surface to be considerably more complex for the IIR MLP model, and for gradient descent update to experience increased difficulty optimizing the function.

[11]The neurons were observed to have gone into saturation, providing a constant output.

## References

Back, A. and Tsoi, A. (1991a). FIR and IIR synapses, a new neural network architecture for time series modelling. *Neural Computation*, 3(3):337–350.

Back, A. D. and Tsoi, A. C. (1991b). Analysis of hidden layer weights in a dynamic locally recurrent network. In Simula, O., editor, *Proceedings International Conference on Artificial Neural Networks, ICANN-91*, volume 1, pages 967–976, Espoo, Finland.

Casdagli, M. (1991). Chaos and deterministic versus stochastic non-linear modelling. *J.R. Statistical Society B*, 54(2):302–328.

Darken, C. and Moody, J. (1991). Note on learning rate schedules for stochastic optimization. In *Neural Information Processing Systems 3*, pages 832–838. Morgan Kaufmann.

de Vries, B. and Principe, J. (1992). The gamma model - a new neural network for temporal processing. *Neural Networks*, 5(4):565–576.

Lang, K. J., Waibel, A. H., and Hinton, G. E. (1990). A time-delay neural network architecture for isolated word recognition. *Neural Networks*, 3:23–43.

Shynk, J. (1989). Adaptive IIR filtering. *IEEE ASSP Magazine*, pages 4–21.
